# Getting lost in space: Large sample analysis of the commute distance

**Ulrike von Luxburg**         **Agnes Radl**
Max Planck Institute for Biological Cybernetics, Tübingen, Germany
{ulrike.luxburg,agnes.radl}@tuebingen.mpg.de

**Matthias Hein**
Saarland University, Saarbrücken, Germany
hein@cs.uni-sb.de

## Abstract

The commute distance between two vertices in a graph is the expected time it takes a random walk to travel from the first to the second vertex and back. We study the behavior of the commute distance as the size of the underlying graph increases. We prove that the commute distance converges to an expression that does not take into account the structure of the graph at all and that is completely meaningless as a distance function on the graph. Consequently, the use of the raw commute distance for machine learning purposes is strongly discouraged for large graphs and in high dimensions. As an alternative we introduce the amplified commute distance that corrects for the undesired large sample effects.

## 1 Introduction

Given an undirected, weighted graph, the commute distance between two vertices $u$ and $v$ is defined as the expected time it takes a random walk starting in vertex $u$ to travel to vertex $v$ and back to $u$. As opposed to the shortest path distance, it takes into account all paths between $u$ and $v$, not just the shortest one. As a rule of thumb, the more paths connect $u$ with $v$, the smaller the commute distance becomes. As a consequence, it supposedly satisfies the following, highly desirable property:

> **Property ($\bigstar$):** Vertices in the same cluster of the graph have a small commute distance, whereas two vertices in different clusters of the graph have a "large" commute distance.

It is because of this property that the commute distance has become a popular choice and is widely used, for example in clustering (Yen et al., 2005), semi-supervised learning (Zhou and Schölkopf, 2004), in social network analysis (Liben-Nowell and Kleinberg, 2003), for proximity search (Sarkar et al., 2008), in image processing (Qiu and Hancock, 2005), for dimensionality reduction (Ham et al., 2004), for graph embedding (Guattery, 1998, Saerens et al., 2004, Qiu and Hancock, 2006, Wittmann et al., 2009) and even for deriving learning theoretic bounds for graph labeling (Herbster and Pontil, 2006, Cesa-Bianchi et al., 2009). One of the main contributions of this paper is to establish that property ($\bigstar$) does not hold in many relevant situations.

In this paper we study how the commute distance (up to a constant factor equivalent to the resistance distance, see below for exact definitions) behaves when the size of the graph increases. We focus on the case of random geometric graphs as this is most relevant to machine learning, but similar results hold for very general classes of graphs under mild assumptions. Denoting by $H_{ij}$ the expected hitting time, by $C_{ij}$ the commute distance between two vertices $v_i$ and $v_j$ and by $d_i$ the degree of

vertex $v_i$ we prove that the hitting times and commute distances can be approximated (up to the constant $\text{vol}(G)$ that denotes the volume of the graph) by

$$\frac{1}{\text{vol}(G)} H_{ij} \approx \frac{1}{d_j} \qquad \text{and} \qquad \frac{1}{\text{vol}(G)} C_{ij} \approx \frac{1}{d_i} + \frac{1}{d_j}.$$

The intuitive reason for this behavior is that if the graph is large, the random walk "gets lost" in the sheer size of the graph. It takes so long to travel through a substantial part of the graph that by the time the random walk comes close to its goal it has already "forgotten" where it started from. For this reason, the hitting time $H_{ij}$ does not depend on the starting vertex $v_i$ any more. It only depends on the inverse degree of the target vertex $v_j$, which intuitively represents the likelihood that the random walk exactly hits $v_j$ once it is in its neighborhood. In this respect it shows the same behavior as the mean return time at $j$ (the mean time it takes a random walk that starts at $j$ to return to its staring point) which is well-known to be $\text{vol}(G) \cdot 1/d_j$ as well.

Our findings have very strong implications:

**The raw commute distance is not a useful distance function on large graphs.** On the negative side, our approximation result shows that contrary to popular belief, the commute distance does not take into account any global properties of the data, at least if the graph is "large enough". It just considers the local density (the degree of the vertex) at the two vertices, nothing else. The resulting large sample commute distance $dist(v_i, v_j) = 1/d_i + 1/d_j$ is completely meaningless as a distance on a graph. For example, all data points have the same nearest neighbor (namely, the vertex with the largest degree), the same second-nearest neighbor (the vertex with the second-largest degree), and so on. In particular, the main motivation to use the commute distance, Property (★), no longer holds when the graph becomes "large enough". Even more disappointingly, computer simulations show that $n$ does not even need to be very large before (★) breaks down. Often, $n$ in the order of 1000 is already enough to make the commute distance very close to its approximation expression (see Section 5 for details). This effect is even stronger if the dimensionality of the underlying data space is large. Consequently, even on moderate-sized graphs, the use of the raw commute distance as a basis for machine learning algorithms should be discouraged.

**Correcting the commute distance.** It has been reported in the literature that hitting times and commute times can be observed to be quite small if the vertices under consideration have a high degree, and that the spread of the commute distance values can be quite large (Liben-Nowell and Kleinberg, 2003, Brand, 2005, Yen et al., 2009). Subsequently, the authors suggested several different methods to correct for this unpleasant behavior. In the light of our theoretical results we can see immediately why the undesired behavior of the commute distance occurs. Moreover, we are able to analyze the suggested corrections and prove which ones are meaningful and which ones not (see Section 4). Based on our theory we suggest a new correction, the amplified commute distance. This is a new distance function that is derived from the commute distance, but avoids its artifacts. This distance function is Euclidean, making it well-suited for machine learning purposes and kernel methods.

**Efficient computation of approximate commute distances.** In some applications the commute distance is not used as a distance function, but for other reasons, for example in graph sparsification (Spielman and Srivastava, 2008) or when computing bounds on mixing or cover times (Aleliunas et al., 1979, Chandra et al., 1989, Avin and Ercal, 2007, Cooper and Frieze, 2009) or graph labeling (Herbster and Pontil, 2006, Cesa-Bianchi et al., 2009). To obtain the commute distance between all points in a graph one has to compute the pseudo-inverse of the graph Laplacian matrix, an operation of time complexity $O(n^3)$. This is prohibitive in large graphs. To circumvent the matrix inversion, several approximations of the commute distance have been suggested in the literature (Spielman and Srivastava, 2008, Sarkar and Moore, 2007, Brand, 2005). Our results lead to a much simpler and well-justified way of approximating the commute distance on large random geometric graphs.

## 2 General setup, definitions and notation

We consider undirected, weighted graphs $G = (V, E)$ with $n$ vertices. We always assume that $G$ is connected and not bipartite. The non-negative weight matrix (adjacency matrix) is denoted by $W := (w_{ij})_{i,j=1,...,n}$. By $d_i := \sum_{j=1}^{n} w_{ij}$ we denote the degree of vertex $v_i$ and $\text{vol}(G) := \sum_{j=1}^{n} d_j$ is the volume of the graph. $D$ denotes the diagonal matrix with diagonal entries $d_1, \ldots, d_n$ and is called the *degree matrix*.

Our main focus in this paper is the class of *random geometric graphs* as it is most relevant to machine learning. Here we are given a sequence of points $X_1, \ldots, X_n$ that has been drawn i.i.d. from some underlying density $p$ on $\mathbb{R}^d$. These points form the vertices $v_1, \ldots, v_n$ of the graph. The edges in the graph are defined such that "neighboring points" are connected: In the $\varepsilon$-*graph* we connect two points whenever their Euclidean distance is less than or equal to $\varepsilon$. In the undirected, *symmetric k-nearest neighbor graph* we connect $v_i$ to $v_j$ if $X_i$ is among the $k$ nearest neighbors of $X_j$ *or* vice versa. In the *mutual k-nearest neighbor graph* we connect $v_i$ to $v_j$ if $X_i$ is among the $k$ nearest neighbors of $X_j$ *and* vice versa. For space constraints we only discuss the case of unweighted graphs in this paper. Our results can be carried over to weighted graphs, in particular to weighted kNN-graphs and Gaussian similarity graphs.

Consider the natural random walk on $G$, that is the random walk with transition matrix $P = D^{-1}W$. The *hitting time* $H_{ij}$ is defined as the expected time it takes a random walk starting in vertex $v_i$ to travel to vertex $v_j$ (with $H_{ii} := 0$ by definition). The *commute distance* (also called commute time) between $v_i$ and $v_j$ is defined as $C_{ij} := H_{ij} + H_{ji}$. Some readers might also know the commute distance under the name *resistance distance*. Here one interprets the graph as an electrical network where the edges represent resistors. The conductance of a resistor is given by the corresponding edge weight. The resistance distance $R_{ij}$ between $i$ and $j$ is defined as the effective resistance between the vertices $i$ and $j$ in the network. It is well known that the resistance distance coincides with the commute distance up to a constant: $C_{ij} = \text{vol}(G) \cdot R_{ij}$. For background reading see Doyle and Snell (1984), Klein and Randic (1993), Xiao and Gutman (2003), Fouss et al. (2006), Bollobás (1998), Lyons and Peres (2010).

For the rest of the paper we consider a probability distribution with density $p$ on $\mathbb{R}^d$. We want to study the behavior of the commute distance between two fixed points $s$ and $t$. We will see that we only need to study the density in a reasonably small region $\mathcal{X} \subset \mathbb{R}^d$ that contains $s$ and $t$. For convenience, let us make the following definition.

**Definition 1 (Valid region)** *Let $p$ be any density on $\mathbb{R}^d$, and $s, t \in \mathbb{R}^d$ be two points with $p(s), p(t) > 0$. We call a connected subset $\mathcal{X} \subset \mathbb{R}^d$ a* valid region *with respect to $s, t$ and $p$ if the following properties are satisfied:*

1. *$s$ and $t$ are interior points of $\mathcal{X}$.*
2. *The density on $\mathcal{X}$ is bounded away from 0, that is for all $x \in \mathcal{X}$ we have that $p(x) \geq p_{\min} > 0$ for some constant $p_{\min}$. Assume that $p_{\max} := \max_{x \in \mathcal{X}} p(x) < \infty$.*
3. *$\mathcal{X}$ has "bottleneck" larger than some value $h > 0$: the set $\{x \in \mathcal{X} : dist(x, \partial\mathcal{X}) > h/2\}$ is connected (here $\partial\mathcal{X}$ denotes the topological boundary of $\mathcal{X}$).*
4. *The boundary of $\mathcal{X}$ is regular in the following sense. We assume that there exist positive constants $\alpha > 0$ and $\varepsilon_0 > 0$ such that if $\varepsilon < \varepsilon_0$, then for all points $x \in \partial\mathcal{X}$ we have $\text{vol}(B_\varepsilon(x) \cap \mathcal{X}) \geq \alpha \, \text{vol}(B_\varepsilon(x))$ (where $\text{vol}$ denotes the Lebesgue volume). Essentially this condition just excludes the situation where the boundary has arbitrarily thin spikes.*

For readability reasons, we are going to state some of our main results using constants $c_i > 0$. These constants are independent of $n$ and the graph connectivity parameter ($\varepsilon$ or $k$, respectively) but depend on the dimension, the geometry of $\mathcal{X}$, and $p$. The values of all constants are determined explicitly in the proofs. They do not coincide across different propositions. For notational convenience, we will formulate all the following results in terms of the resistance distance. To obtain the results for the commute distance one just has to multiply by factor $\text{vol}(G)$.

## 3 Convergence of the resistance distance on random geometric graphs

In this section we present our theoretical main results for random geometric graphs. We show that on this type of graph, the resistance distance $R_{ij}$ converges to the trivial limit $1/d_i + 1/d_j$. For space constraints we only formulate these results for unweighted kNN and $\varepsilon$-graphs. Similar results also hold for weighted variants of these graphs and for Gaussian similarity graphs.

**Theorem 2 (Resistance distance on kNN-graphs)** *Fix two points $X_i$ and $X_j$. Consider a valid region $\mathcal{X}$ with respect to $X_i$ and $X_j$ with bottleneck $h$ and density bounds $p_{\min}$ and $p_{\max}$. Assume that $X_i$ and $X_j$ have distance at least $h$ from the boundary of $\mathcal{X}$ and that $(k/n)^{1/d}/2p_{\max} \leq h$.*

*Then there exist constants $c_1, \ldots, c_5 > 0$ such that with probability at least $1 - c_1 n \exp(-c_2 k)$ the resistance distance on both the symmetric and the mutual kNN-graph satisfies*

$$\left| k R_{ij} - \left( \frac{k}{d_i} + \frac{k}{d_j} \right) \right| \leq \begin{cases} c_4 \frac{1}{k} \left( \log(n/k) + (k/n)^{1/3} + 1 \right) & \text{if } d = 3 \\ c_5 \frac{1}{k} & \text{if } d > 3 \end{cases}$$

*The probability converges to 1 if $n \to \infty$ and $k/\log(n) \to \infty$. The rhs of the deviation bound converges to 0 as $n \to \infty$, if $k \to \infty$ and $k/\log(n/k) \to \infty$ in case $d = 3$, and if $k \to \infty$ in case $d > 3$. Under these conditions, if the density $p$ is continuous and if additionally $k/n \to 0$, then $k R_{ij} \to 2$ in probability.*

**Theorem 3 (Resistance distance on $\varepsilon$-graphs)** *Fix two points $X_i$ and $X_j$. Consider a valid region $\mathcal{X}$ with respect to $X_i$ and $X_j$ with bottleneck $h$ and density bounds $p_{\min}$ and $p_{\max}$. Assume that $X_i$ and $X_j$ have distance at least $h$ from the boundary of $\mathcal{X}$ and that $\varepsilon \leq h$. Then there exist constants $c_1, \ldots, c_6 > 0$ such that with probability at least $1 - c_1 n \exp(-c_2 n \varepsilon^d) - c_3 \exp(-c_4 n \varepsilon^d)/\varepsilon^d$ the resistance distance on the $\varepsilon$-graph satisfies*

$$\left| n \varepsilon^d R_{ij} - \left( \frac{n \varepsilon^d}{d_i} + \frac{n \varepsilon^d}{d_j} \right) \right| \leq \begin{cases} c_5 \frac{\log(1/\varepsilon) + \varepsilon + 1}{n \varepsilon^3} & \text{if } d = 3 \\ c_6 \frac{1}{n \varepsilon^d} & \text{if } d > 3 \end{cases}$$

*The probability converges to 1 if $n \to \infty$ and $n \varepsilon^d / \log(n) \to \infty$. The rhs of the deviation bound converges to 0 as $n \to \infty$, if $n \varepsilon^3 / \log(1/\varepsilon) \to \infty$ in case $d = 3$, and if $n \varepsilon^d \to \infty$ in case $d > 3$. Under these conditions, if the density $p$ is continuous and if additionally $\varepsilon \to 0$, then*

$$n \varepsilon^d R_{ij} \to \frac{1}{\eta_d p(X_i)} + \frac{1}{\eta_d p(X_j)} \quad \text{in probability.}$$

Let us discuss the theorems en bloc. We start with a couple of technical remarks. Note that to achieve the convergence of the resistance distance we have to rescale it appropriately (for example, in the $\varepsilon$-graph we scale by a factor of $n \varepsilon^d$). Our rescaling is exactly chosen such that the limit expressions are finite, positive values. Scaling by any other factor in terms of $n$, $\varepsilon$ or $k$ either leads to divergence or to convergence to zero.

The convergence conditions on $n$ and $\varepsilon$ (or $k$, respectively) are the ones to be expected for random geometric graphs. They are satisfied as soon as the degrees are of the order $\log(n)$ (for smaller degrees, the graphs are not connected anyway, see e.g. Penrose, 1999). Hence, our results hold for sparse as well as for dense connected random geometric graphs.

The valid region $\mathcal{X}$ has been introduced for technical reasons. We need to operate in such a region in order to be able to control the behavior of the graph, e.g. the average degrees. The assumptions on $\mathcal{X}$ are the standard assumptions used in the random geometric graph literature. In our setting, we have the freedom of choosing $\mathcal{X} \subset \mathbb{R}^d$ as we want. In order to obtain the tightest bounds one should aim for a valid $\mathcal{X}$ that has a wide bottleneck and a high minimal density.

More generally, results about the convergence of the commute distance to $1/d_i + 1/d_j$ can also be proved for other kinds of graphs such as graphs with given expected degrees and even for power law graphs, under the assumption that the minimal degree in the graph slowly increases with $n$. Details are beyond the scope of this paper.

**Proof outline of Theorems 2 and 3** (full proofs are presented in the supplementary material). Consider two fixed vertices $s$ and $t$ in a connected graph and consider the graph as an electrical network where each edge has resistance 1. By the electrical laws, resistances in series add up, that is for two resistances $R_1$ and $R_2$ in series we get the overall resistance $R = R_1 + R_2$. Resistances in parallel lines satisfy $1/R = 1/R_1 + 1/R_2$. Now consult the situation in Figure 1. Consider the vertex $s$ and all edges from $s$ to its $d_s$ neighbors. The resistance "spanned" by these $d_s$ parallel edges satisfies $1/R = \sum_{i=1}^{d_s} 1$, that is $R = 1/d_s$. Similarly for $t$. Between the neighbors of $s$ and the ones of $t$ there are very many paths. It turns out that the contribution of these paths to the resistance is negligible (essentially, we have so many wires between the two neighborhoods that electricity can flow nearly freely). So the overall effective resistance between $s$ and $t$ is dominated by the edges adjacent to $i$ and $j$ with contributions $1/d_s + 1/d_t$.

Providing a clean mathematical proof for this argument is quite technical. Our proof is based on Corollary 6 in Section IX.2 of Bollobás (1998) that states that the resistance distance beween two

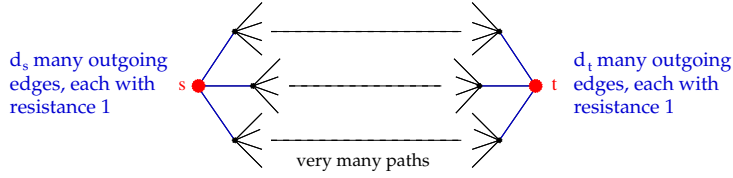

<small>d<sub>s</sub> many outgoing edges, each with resistance 1</small>

<small>d<sub>t</sub> many outgoing edges, each with resistance 1</small>

Figure 1: *Intuition for the proof of Theorems 2 and 3. See text for details.*

fixed vertices $s$ and $t$ can be expressed as

$$R_{st} = \inf\left\{ \sum_{e \in E} u_e^2 \,\Big|\, u = (u_e)_{e \in E} \text{ unit flow from } s \text{ to } t \right\}.$$

To apply this theorem one has to construct a flow that spreads "as widely as possible" over the whole graph. Counting edges and adding up resistances then leads to the desired results. Details are fiddly and can be found in the supplementary material.  ☺

## 4  Correcting the resistance distance

Obviously, the large sample resistance distance $R_{ij} \approx 1/d_i + 1/d_j$ is completely meaningless as a distance on a graph. The question we want to discuss in this section is whether there is a way to correct the commute distance such that this unpleasant large sample effect does not occur. Let us start with some references to the literature. It has been observed in several empirical studies that the commute distances are quite small if the vertices under consideration have a high degree, and that the spread of the commute distance values can be quite large. Our theoretical results immediately explain this behavior: if the degrees are large, then $1/d_i + 1/d_j$ is very small. And compared to the "spread" of $d_i$, the spread of $1/d_i$ can be enormous.

Several heuristics have been suggested to solve this problem. Liben-Nowell and Kleinberg (2003) suggest to correct the hitting times by simply multiplying by the degrees. For the commute distance, this leads to the suggested correction of $C_{LNK}(i,j) := d_j H_{ij} + d_i H_{ji}$. Even though we did not prove it explicitly in our paper, the convergence results for the commute time also hold for the individual hitting times. Namely, hitting time $H_{ij}$ can be approximated by $vol(G)/d_j$. These theoretical results immediately show that the correction $C_{LNK}$ is not useful, at least if we consider the absolute values. For large graphs, it simply has the effect of normalizing all hitting times to $\approx 1$, leading to $C_{LNK} \approx 2$. However, we believe that the *ranking* introduced by this distance function still contains useful information about the data. The reason is that while the first order terms dominate the absolute value and converge to two, the second order terms introduce some "variation around two", and this variation might encode the cluster structure.

Yen et al. (2009) exploit the well-known fact that the commute distance is Euclidean and its kernel matrix coincides with the Moore-Penrose inverse $L^+$ of the graph Laplacian matrix. The authors now apply a sigmoid transformation to $L^+$ and consider $K_{\text{Yen}}(i,j) = 1/(1 + \exp(-l_{ij}^+/\sigma))$ for some contant $\sigma$. The idea is that the sigmoid transformation reduces the spread of the distance (or similarity) values. However, this is an ad-hoc approach that has the disadvantage that the resulting "kernel" $K_{\text{Yen}}$ is not positive definite.

A third correction has been suggested in Brand (2005). As Yen et al. (2009) he considers the kernel matrix that corresponds to the commute distance. But instead of applying a sigmoid transformation he centers and normalizes the kernel matrix in the feature space. This leads to the corrected kernel

$$K_{\text{Brand}}(i,j) = \bar{K}_{ij}/\sqrt{\bar{K}_{ii}\bar{K}_{jj}} \qquad \text{with} \qquad 2\bar{K}_{ij} = -R_{ij} + \tfrac{1}{n}\sum_{k=1}^n (R_{ik} + R_{kj}) - \tfrac{1}{n^2}\sum_{k,l=1}^n R_{kl}.$$

One the first glance it is surprising that using the centered and normalized kernel instead of the commute distance should make any difference. However, whenever one takes a Euclidean distance function of the form $\text{dist}(i,j) = s_{ij} + u_i + u_j - 2\delta_{ij}u_i$ and computes the corresponding *centered* kernel matrix, one obtains

$$K_{ij} = K_{ij}^s + 2\delta_{ij}u_i - \frac{2}{n}(u_i + u_j) + \frac{2}{n^2}\sum_{r=1}^n u_r, \tag{1}$$

where $K^s$ is the kernel matrix induced by $s$. Thus the off-diagonal terms are still influenced by $u_i$ but with a decaying factor $\frac{1}{n}$ compared to the diagonal. Even though this is no longer the case after normalization (because for the normalization the diagonal terms are important, and these terms still depend on the $d_i$), we believe that this is the key to why Brand's kernel is useful.

What would be a suitable correction based on our theoretical results? The proof of our main theorems shows that the edges adjacent to $i$ and $j$ completely dominate the behavior of the resistance distance: they are the "bottleneck" of the flow, and their contribution $1/d_i + 1/d_j$ dominates all the other terms. The interesting information about the global topology of the graph is contained in the remainder terms $S_{ij} = R_{ij} - 1/d_i - 1/d_j$, which summarize the flow contributions of all other edges in the graph. We believe that the key to obtaining a good distance function is to remove the influence of the $1/d_i$ terms and "amplify" the influence of the general graph term $S_{ij}$. This can be achieved by either using the off-diagonal terms of the pseudo-inverse graph Laplacian $L^{\dagger}$ while ignoring its diagonal, or by building a distance function based on the remainder terms $S_{ij}$ directly. We choose the second option and propose the following new distance function. We define the **amplified commute distance** as $C_{\mathrm{amp}}(i,j) = S_{ij} + u_{ij}$ with $S_{ij} = R_{ij} - 1/d_i - 1/d_j$ and $u_{ij} = 2w_{ij}/d_i d_j - w_{ii}/d_i^2 - w_{jj}/d_j^2$. Of course we set $C_{\mathrm{amp}}(i,i) = 0$ for all $i$.

**Proposition 4 (Amplified commute distance is Euclidean)** *The matrix $D$ with entries $d_{ij} = C_{amp}(i,j)^{1/2}$ is a Euclidean distance matrix.*

*Proof outline.* In preliminary work we show that the remainder terms can be written as $S_{ij} = \langle (e_i - e_j), B(e_i - e_j) \rangle - u_{ij}$ where $e_i$ denotes the $i$-th unit vector and $B$ is a positive definite matrix (see the proof of Proposition 2 in von Luxburg et al., 2010). This implies the desired statement. ☺

Additionally to being a Euclidean distance, the amplified commute distance has a nice limit behavior. When $n \to \infty$ the terms $u_{ij}$ are dominated by the terms $S_{ij}$, hence all that is left are the "interesting terms" $S_{ij}$. For all practical purposes, one should use the kernel induced by the amplified commute distance and center and normalize it. In formulas, the **amplified commute kernel** is

$$K_{\mathrm{amp}}(i,j) := \bar{K}_{ij}/\sqrt{\bar{K}_{ii}\bar{K}_{jj}} \qquad \text{with} \qquad \bar{K} = (I - \frac{1}{n}\mathbb{1}\mathbb{1}')C_{\mathrm{amp}}(I - \frac{1}{n}\mathbb{1}\mathbb{1}') \qquad (2)$$

(where $I$ is the identity matrix, $\mathbb{1}$ the vector of all ones, and $C_{\mathrm{amp}}$ the amplified commute distance matrix). The next section shows that the kernel $K_{\mathrm{amp}}$ works very nicely in practice.

Note that the correction by Brand and our amplified commute kernel are very similar, but not identical with each other. The off-diagonal terms of both kernels are very close to each other, see Equation (1), that is if one is only interested in a ranking based on similarity values, both kernels behave similarly. However, an important difference is that the diagonal terms in the Brand kernel are way bigger than the ones in the amplified kernel (using our convergence techniques one can show that the Brand kernel converges to an identity matrix, that is the diagonal completely dominates the off-diagonal terms). This might lead to the effect that the Brand kernel behaves worse than our kernel with algorithms like the SVM that do not ignore the diagonal of the kernel.

## 5   Experiments

Our first set of experiments considers the question how fast the convergence of the commute distance takes place in practice. We will see that already for relatively small data sets, a very good approximation takes place. This means that the problems of the raw commute distance already occur for small sample size. Consider the plots in Figure 2. They report the maximal relative error defined as $\max_{ij} |R_{ij} - 1/d_i - 1/d_j|/R_{ij}$ and the corresponding mean relative error on a $\log_{10}$-scale. We show the results for $\varepsilon$-graphs, unweighted kNN graphs and Gaussian similarity graphs (fully connected weighted graphs with edge weights $\exp(\|x_i - x_j\|^2/\sigma^2)$). In order to be able to plot all results in the same figure, we need to match the parameters of the different graphs. Given some value $k$ for the kNN-graph we thus set the values of $\varepsilon$ for the $\varepsilon$-graph and $\sigma$ for the Gaussian graph to be equal to the maximal $k$-nearest neighbor distance in the data set.

*Sample size.* Consider a set of points drawn from the uniform distribution on the unit cube in $\mathbb{R}^{10}$. As can be seen in Figure 2 (first plot), the maximal relative error decreases very fast with increasing sample size. Note that already for small sample sizes the maximal deviations get very small.
*Dimension.* A result that seems surprising at first glance is that the maximal deviation decreases

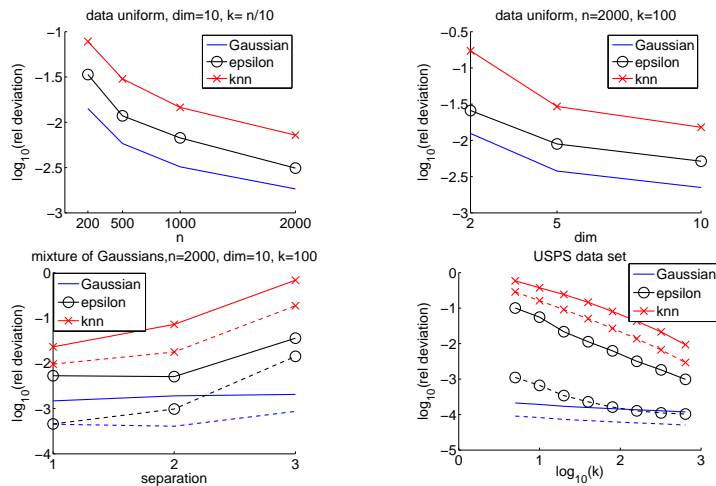

Figure 2: *Relative deviations between true and approximate commute distances. Solid lines show the maximal relative deviations, dashed lines the mean relative deviations. See text for details.*

as we increase the dimension, see Figure 2 (second plot). The intuitive explanation is that in higher dimensions, geometric graphs mix faster as there exist more "shortcuts" between the two sides of the point cloud. Thus, the random walk "forgets faster" where it started from.

*Clusteredness.* The deviation gets worse if the data has a more pronounced cluster structure. Consider a mixture of two Gaussians in $\mathbb{R}^{10}$ with unit variances and the same weight on both components. We call the distance between the centers of the two components the separation. In Figure 2 (third plot) we show both the maximum relative errors (solid lines) and mean relative errors (dashed lines). We can clearly see that with increasing separation, the deviation increases.

*Sparsity.* The last plot of Figure 2 shows the relative errors for increasingly dense graphs, namely for increasing parameter $k$. Here we used the well-known USPS data set of handwritten digits (9298 points in 256 dimensions). We plot both the maximum relative errors (solid lines) and mean relative errors (dashed lines). We can see that the errors decrease the denser the graph gets. Again this is due to the fact that the random walk mixes faster on denser graphs. Note that the deviations are extremely small on this real-world data set.

In a second set of experiments we compare the different corrections of the raw commute distance. To this end, we built a kNN graph of the whole USPS data set (all 9298 points, $k = 10$), computed the commute distance matrix and the various corrections. The resulting matrices are shown in Figure 3 (left part) as heat plots. In all cases, we only plot the off-diagonal terms. We can see that as predicted by theory, the raw commute distance does not identify the cluster structure. However, the cluster structure is still visible in the kernel corresponding to the commute distance, the pseudo-inverse graph Laplacian $L^{\dagger}$. The reason is that the diagonal of this matrix can be approximated by $(1/d_1, ...., 1/d_n)$, whereas the off-diagonal terms encode the graph structure, but on a much smaller scale than the diagonal. In our heat plots, all four corrections of the graph Laplacian show the cluster structure to a certain extent (the correction by LNK to a small extent, the corrections by Brand, Yen and us to a bigger extent).

A last experiment evaluates the performance of the different distances in a semi-supervised learning task. On the whole USPS data set, we first chose some random points to be labeled. Then we classified the unlabeled points by the $k$-nearest neighbor classifier based on the distances to the labeled data points. For each classifier, $k$ was chosen by 10-fold cross-validation among $k \in \{1, ..., 10\}$. The experiment was repeated 10 times. The mean results can be seen in Figure 3 (right figure). As baseline we also report results based on the standard Euclidean distance between the data points. As predicted by theory, we can see that the raw commute distance performs extremely poor. The Euclidean distance behaves reasonably, but is outperformed by all corrections of the commute distance. This shows first of all that using the graph structure does help over the basic Euclidean distance. While the naive correction by LNK stays close to the Euclidean distance, the three corrections by Brand, Yen and us virtually lie on top of each other and outperform the

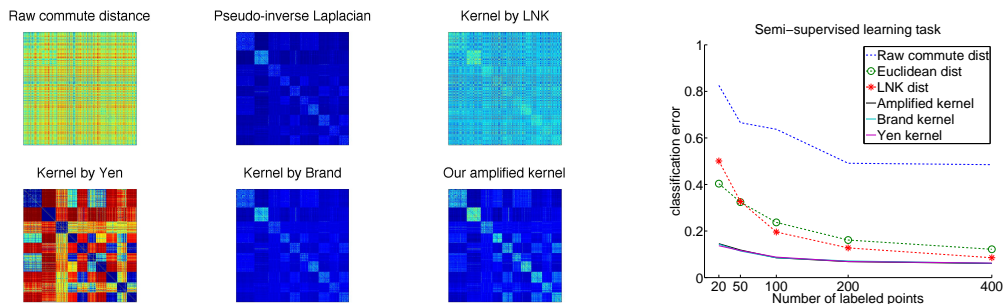

Figure 3: Figures on the left: *Distances and kernels based on a kNN graph between all 9298 USPS points (heat plots, off-diagonal terms only): exact resistance distance, pseudo-inverse graph Laplacian $L^{\dagger}$; kernels corresponding to the corrections by LNK, Yen, Brand, and our amplified $K_{amp}$.* Figure on the right: *Semi-supervised learning results based on the different distances and kernels. The last three lines corresponding to the amplified, Brand and Yen kernel lie on top of each other.*

other methods by a large margin.

We conclude with the following tentative statements. We believe that the correction by LNK is "a bit too naive", whereas the corrections by Brand, Yen and us "tend to work" in a ranking based setting. Based on our simple experiments it is impossible to judge which out of these candidates is "the best one". We are not too fond of Yen's correction because it does not lead to a proper kernel. Both Brand's and our kernel converge to (different) limit functions. So far we do not know the theoretical properties of these limit functions and thus cannot present any theoretical reason to prefer one over the other. However, we think that the diagonal dominance of the Brand kernel can be problematic.

## 6 Discussion

In this paper we have proved that the commute distance on random geometric graphs can be approximated by a very simple limit expression. Contrary to intuition, this limit expression no longer takes into account the cluster structure of the graph, nor any other global property (such as distances in the underlying Euclidean space). Both our theoretical bounds and our simulations tell the same story: the approximation gets better if the data is high-dimensional and not extremely clustered, both of which are standard situations in machine learning. This shows that the use of the raw commute distance for machine learning purposes can be problematic. However, the structure of the graph can be recovered by certain corrections of the commute distance. We suggest to use either the correction by Brand (2005) or our own amplified commute kernel from Section 4. Both corrections have a well-defined, non-trivial limit and perform well in experiments.

The intuitive explanation for our result is that as the sample size increases, the random walk on the sample graph "gets lost" in the sheer size of the graph. It takes so long to travel through a substantial part of the graph that by the time the random walk comes close to its goal it has already "forgotten" where it started from. Stated differently: the random walk on the graph has mixed before it hits the desired target vertex. On a higher level, we expect that the problem of "getting lost" not only affects the commute distance, but many other methods where random walks are used in a naive way to explore global properties of a graph. For example, the results in Nadler et al. (2009), where artifacts of semi-supervised learning in the context of many unlabeled points are studied, seem strongly related to our results. In general, we believe that one has to be particularly careful when using random walk based methods for extracting global properties of graphs in order to avoid getting lost and converging to meaningless results.

# References

R. Aleliunas, R. Karp, R. Lipton, L. Lovász, and C. Rackoff. Random walks, universal traversal sequences, and the complexity of maze problems. In *FOCS*, 1979.

C. Avin and G. Ercal. On the cover time and mixing time of random geometric graphs. *Theor. Comput. Sci*, 380(1-2):2–22, 2007.

B. Bollobás. *Modern Graph Theory*. Springer, 1998.

M. Brand. A random walks perspective on maximizing satisfaction and profit. In *SDM*, 2005.

N. Cesa-Bianchi, C. Gentile, and F. Vitale. Fast and optimal prediction on a labeled tree. In *COLT*, 2009.

A. Chandra, P. Raghavan, W. Ruzzo, R. Smolensky, and P. Tiwari. The electrical resistance of a graph captures its commute and cover times. In *STOC*, 1989.

C. Cooper and A. Frieze. The cover time of random geometric graphs. In *SODA*, 2009.

P. G. Doyle and J. L. Snell. *Random walks and electric networks*. Mathematical Association of America, Washington, DC, 1984.

F. Fouss, A. Pirotte, J.-M. Renders, and M. Saerens. A novel way of computing dissimilarities between nodes of a graph, with application to collaborative filtering and subspace projection of the graph nodes. Technical Report IAG WP 06/08, Université catholique de Louvain, 2006.

S. Guattery. Graph embeddings, symmetric real matrices, and generalized inverses. Technical report, Institute for Computer Applications in Science and Engineering, NASA Research Center, 1998.

J. Ham, D. D. Lee, S. Mika, and B. Schölkopf. A kernel view of the dimensionality reduction of manifolds. In *ICML*, 2004.

M. Herbster and M. Pontil. Prediction on a graph with a perceptron. In *NIPS*, 2006.

D. Klein and M. Randic. Resistance distance. *Journal of Mathematical Chemistry*, 12:81–95, 1993.

D. Liben-Nowell and J. Kleinberg. The link prediction problem for social networks. In *CIKM*, 2003.

R. Lyons and Y. Peres. Probability on trees and networks. *Book in preparation, available online on the webpage of Yuval Peres*, 2010.

B. Nadler, N. Srebro, and X. Zhou. Statistical analysis of semi-supervised learning: The limit of infinite unlabelled data. In *NIPS*, 2009.

M. Penrose. A strong law for the longest edge of the minimal spanning tree. *Ann. of Prob.*, 27(1): 246 – 260, 1999.

H. Qiu and E. R. Hancock. Image segmentation using commute times. In *BMVC*, 2005.

H. Qiu and E. R. Hancock. Graph embedding using commute time. *S+SSPR 2006*, pages 441–449, 2006.

M. Saerens, F. Fouss, L. Yen, and P. Dupont. The principal components analysis of a graph, and its relationships to spectral clustering. In *ECML*, 2004.

P. Sarkar and A. Moore. A tractable approach to finding closest truncated-commute-time neighbors in large graphs. In *UAI*, 2007.

P. Sarkar, A. Moore, and A. Prakash. Fast incremental proximity search in large graphs. In *ICML*, 2008.

D. Spielman and N. Srivastava. Graph sparsification by effective resistances. In *STOC*, 2008.

U. von Luxburg, A. Radl, and M. Hein. Hitting times, commute distances and the spectral gap in large random geometric graphs. *Preprint available at Arxiv*, March 2010.

D. M. Wittmann, D. Schmidl, F. Blöchl, and F. J. Theis. Reconstruction of graphs based on random walks. *Theoretical Computer Science*, 2009.

W. Xiao and I. Gutman. Resistance distance and Laplacian spectrum. *Theoretical Chemistry Accounts*, 110:284–298, 2003.

L. Yen, D. Vanvyve, F. Wouters, F. Fouss, M. Verleysen, and M. Saerens. Clustering using a random walk based distance measure. In *ESANN*, 2005.

L. Yen, F. Fouss, C. Decaestecker, P. Francq, and M. Saerens. Graph nodes clustering based on the commute-time kernel. *Advances in Knowledge Discovery and Data Mining*, pages 1037–1045, 2009.

D. Zhou and B. Schölkopf. Learning from Labeled and Unlabeled Data Using Random Walks. In *DAGM*, 2004.

